# Bayesian Model of Behaviour in Economic Games

**Debajyoti Ray**
Computation and Neural Systems
California Institute of Technology
Pasadena, CA 91125. USA
dray@caltech.edu

**Brooks King-Casas**
Computational Psychiatry Unit
Baylor College of Medicine.
Houston, TX 77030. USA
bkcasas@cpu.bcm.tmc.edu

**P. Read Montague**
Human NeuroImaging Lab
Baylor College of Medicine.
Houston, TX 77030. USA
montague@hnl.bcm.tmc.edu

**Peter Dayan**
Gatsby Computational Neuroscience Unit
University College London
London. WC1N 3AR. UK
dayan@gatsby.ucl.ac.uk

## Abstract

Classical game theoretic approaches that make strong rationality assumptions have difficulty modeling human behaviour in economic games. We investigate the role of finite levels of iterated reasoning and non-selfish utility functions in a Partially Observable Markov Decision Process model that incorporates game theoretic notions of interactivity. Our generative model captures a broad class of characteristic behaviours in a multi-round Investor-Trustee game. We invert the generative process for a recognition model that is used to classify 200 subjects playing this game against randomly matched opponents.

## 1   Introduction

Trust tasks such as the Dictator, Ultimatum and Investor-Trustee games provide an empirical basis for investigating social cooperation and reciprocity [11]. Even in completely anonymous settings, human subjects show rich patterns of behavior that can be seen in terms of such personality concepts as charity, envy and guilt. Subjects also behave as if they model these aspects of their partners in games, for instance acting to avoid being taken advantage of. Different subjects express quite different personalities, or types, and also have varying abilities at modelling their opponents.

The burgeoning interaction between economic psychology and neuroscience requires formal treatments of these issues. From the perspective of neuroscience, such treatments can provide a precise quantitative window into neural structures involved in assessing utilties of outcomes, capturing risk and probabilities associated with interpersonal interactions, and imputing intentions and beliefs to others. In turn, evidence from brain responses associated with these factors should elucidate the neural algorithms of complex interpersonal choices, and thereby illuminate economic decision-making.

Here, we consider a sequence of paradigmatic trust tasks that have been used to motivate a variety of behaviorally-based economic models. In brief, we provide a formalization in terms of partially observable Markov decision processes, approximating type-theoretic Bayes-Nash equilibria [8] using finite hierarchies of belief, where subjects' private types are construed as parameters of their inequity averse utility functions [2]. Our inference methods are drawn from machine learning.

Figure 1a shows a simple one-round trust game. In this, an Investor is paired against a randomly assigned Trustee. The Investor can either choose a safe option with a low payoff for both, or take a risk and pass the decision to the Trustee who can either choose to defect (and thus keep more for herself) or choose the fair option that leads to more gains for both players (though less profitable

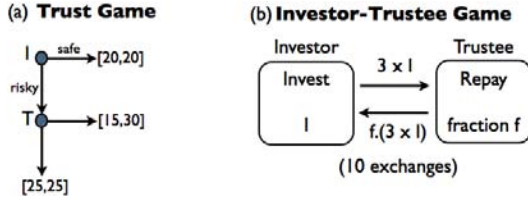

Figure 1: (a) In a simple Trust game, the Investor can take a safe option with a payoff of $[Investor=20,Trustee=20] (i.e. the Investor gets $20 and the Trustee gets $20). The game ends if the Investor chooses the safe option; alternatively, he can pass the decision to the Trustee. The Trustee can now choose a fair option $[25,25] or choose to defect $[15,30]. (b) In the multi-round version of the Trust game, the Investor gets $20 dollars at every round. He can invest any (integer) part; this quantity is trebled on the way to the Trustee. In turn, she has the option of repaying any (integer) amount of her resulting allocation to the Investor. The game continues for 10 rounds.

for herself alone than if she defected). Figure 1b shows the more sophisticated game we consider, namely a multi-round, sequential, version of the Trust game [15].

The fact that even in a purely anonymized setting, Investors invest at all, and Trustees reciprocate at all in games such as that of figure 1a, is a challenge to standard, money-maximizing doctrines (which expect to find the Nash equilibrium where neither happens), and pose a problem for modeling. One popular strategy is to retain the notion that subjects attempt to optimize their utilities, but to include in these utilities social factors that penalize cases in which opponents win either more (crudely *envy*, parameterized by $\alpha$) or less (*guilt*, parameterized by $\beta$) than themselves [2]. One popular Inequity-Aversion utility function [2] characterizes player $i$ by the *type* $T_i = (\alpha_i, \beta_i)$ of her utility function:

$$U(\alpha_i, \beta_i) = x_i - \alpha_i \max\{(x_j - x_i), 0\} - \beta_i \max\{(x_i - x_j), 0\} \tag{1}$$

where $x_i$, $x_j$ are the amounts received by players $i$ and $j$ respectively.

In the multi-round version of figure 1b, reputation formation comes into play [15]. Investors have the possibility of gaining higher rewards from giving money to the Trustee; and, at least until the final round, the Trustee has an incentive to maintain a reputation of trustworthiness in order to coax the Investor to offer more (against any Nash tendencies associated with solipsistic utility functions). Social utility functions such as that of equation 1 mandate probing, belief manipulation and the like.

We cast such tasks as Bayesian Games. As in the standard formulation [8], players know their own types but not those of their opponents; dyads are thus playing games of incomplete information. A player also has prior beliefs about their opponent that are updated in a Bayesian manner after observing the opponent's actions. Their own actions also influence their opponent's beliefs. This leads to an infinite hierarchy of beliefs: what the Trustee thinks of the Investor; what the Trustee thinks the Investor thinks of him; what the Trustee thinks the Investor thinks the Trustee thinks of her; and so on. If players have common prior beliefs over the possible types in the game, and this prior is common knowledge, then (at least one) subjective equilibrium known as the Bayes-Nash Equilibrium (BNE), exists [8]. Algorithms to compute BNE solutions have been developed but, in the general case, are NP-hard [6] and thus infeasible for complex multi-round games [9].

One obvious approach to this complexity is to consider finite rather than infinite belief hierarchies. This has both theoretical and empirical support. First, a finite hierarchy of beliefs can provably approximate the equilibrium solution that arises in an infinite belief hierarchy arbitrarily closely [10], an idea that has indeed been employed in practice to compute equilibria in a multi-agent setting [5]. Second, based on a whole wealth of games such as the p-Beauty game [11], it has been suggested that human subjects only employ a very restricted number of steps of strategic thinking. According to cognitive hierarchy theory, a celebrated account of this, this number is on average a mere 1.5 [13].

In order to capture the range of behavior exhibited by subjects in these games, we built a finite belief hierarchy model, using inequity averse utility functions in the context of a partially observable hidden Markov model of the ignorance each subject has about its opponent's type and in the light of sequential choice. We used inference strategies from machine learning to find approximate solutions to this model. In this paper, we use this generative model to investigate the qualitative classes of behaviour that can emerge in these games.

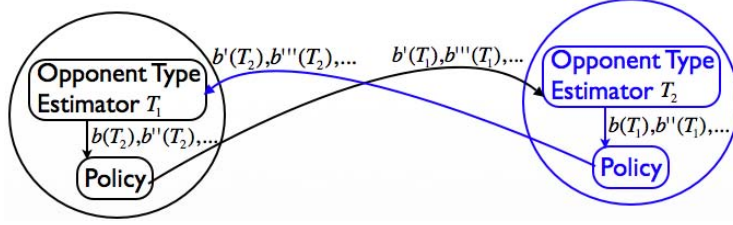

Figure 2: Each player's decision-making requires solving a POMDP, which involves solving the opponent's POMDP. Higher order beliefs are required as each player's action influences the opponent's beliefs which in turn influence their policy.

## 2   Partially Observable Markov Games

As in the framework of Bayesian games, player $i$'s inequity aversion type $T_i = (\alpha_i, \beta_i)$ is known to it, but not to the opponent. Player $i$ does have a prior distribution over the type of the other player $j$, $b_i^{(0)}(T_j)$; and, if suitably sophisticated, can also have higher-order priors over the whole hierarchy of recursive beliefs about types. We denote the collection of priors as $\vec{b}_i^{(0)} = \{b_i^{(0)}, b_i^{(0)'}, b_i^{(0)''}, ...\}$. Play proceeds sequentially, with player $i$ choosing action $a_i^{(t)}$ at time $t$ according to the expected future value of this choice. In this (hidden) Markovian setting, this value, called a $\mathcal{Q}$-value depends on the stage (given the finite horizon), the current beliefs of the player $\vec{b}_i^{(t)}$ (which are sufficient statistics for the past observations), and the policies $P(a_i^{(t)} = a | \mathcal{D}^{(t)})$ (which depend on the observations $\mathcal{D}^{(t)}$) of both players up to time $t$:

$$Q_i^{(t)}(\vec{b}_i^{(t)}, a_i^{(t)}) = U_i^{(t)}(\vec{b}_i^{(t)}, a_i^{(t)}) +$$
$$\sum_{a_j^{(t)} \in A_j^{(t)}} P(a_j^{(t)} | \{\mathcal{D}^{(t)}, a_i^{(t)}\}) \sum_{a_i^{(t+1)} \in A_i^{(t+1)}} Q_i^{(t+1)}(\vec{b}_i^{(t+1)}, a_i^{(t+1)}) P(a_{i+1}^{(t+1)} | \{\mathcal{D}^{(t)}, a_i^{(t)}, a_j^{(t)}\}) \quad (2)$$

where we arbitrarily *define* the softmax policy,

$$P(a_i^{(t)} = a | \mathcal{D}^{(t)}) = \exp\left(\phi Q_i^{(t)}(\vec{b}_i^{(t)}, a)\right) / \sum_b \exp\left(\phi Q_i^{(t)}(\vec{b}_i^{(t)}, b)\right) \quad (3)$$

akin to Quantal Response Equilibrium [12], which depends on player $i$'s beliefs about player $j$, which are, in turn, updated using Bayes' rule based on the likelihood function $P(a_j^{(t)} | \{\mathcal{D}^{(t)}, a_i^{(t)}\})$

$$b_i^{(t+1)}(T_j) = P(T_j | a_j^{(t)}, a_i^{(t)}, b_i^{(t)}) = P(T_j, a_i^{(t)}, a_j^{(t)}) \sum_{T_j'} b_i^{(t)}(T_j') / P(a_j^{(t)} | a_i^{(t)}, b_i^{(t)}) \quad (4)$$

switching between history-based ($\mathcal{D}^t$) and belief-based ($b_i^{(t)}(T_j)$) representations. Given the interdependence of beliefs and actions, we expect to see probing (to find out the type and beliefs of one's opponent) and belief manipulation (being nice now to take advantage of one's opponent later).

If the other player's decisions are assumed to emerge from equivalent softmax choices, then for the subject to calculate this likelihood, they must also solve their opponent's POMDP. This leads to an infinite recursion (illustrated in fig. 2). In order to break this, we assume that each player has $k$ levels of strategic thinking as in the Cognitive Hierarchy framework [13]. Thus each $k$-level player assumes that his opponent is a $k-1$-level player. At the lowest level of the recursion, the 0-level player uses a simple likelihood to update their opponent's beliefs.

The utility $U_i^{(t)}(a_i^{(t)})$ is calculated at every round for each player $i$ for action $a_i^{(t)}$ by marginalizing over the current beliefs $b_i^{(t)}$. It is extremely challenging to compute with belief states, since they are probability distributions, and are therefore continuous-valued rather than discrete. To make this computationally reasonable, we discretize the values of the types. As an example, if there are only two types for a player the belief state, which is a continuous probability distribution over the interval

$[0, 1]$ is discretized to take K values $b_{i1} = 0, \ldots, b_{iK} = 1$. The utility of an action is obtained by marginalizing over the beliefs as:

$$U_i^{(t)}(a_i^{(t)}) = \sum_{k=1:K} b_{ik} Q_i^{(t)}(b_{ik}^{(t)}, a_i^{(t)}) \tag{5}$$

Furthermore, we solve the resulting POMDP using a mixture of explicit expansion of the tree from the current start point to three stages ahead, and a stochastic, particle-filter-based scheme (as in [7]), from four stages ahead to the end of the game.

One characteristic of this explicit process model, or algorithmic approach, is that it is possible to consider what happens when the priors of the players differ. In this case, as indeed also for the case of only a finite belief hierarchy, there is typically no formal Bayes-Nash equilibrium. We also verified our algorithm against the QRE and BNE solutions provided by GAMBIT ([14]) on a 1 and 2 round Trust game for $k = 1, 2$ respectively. However unlike the BNE solution in the extensive form game, our algorithm gives rise to belief manipulation and effects at the end of the game.

## 3  Generative Model for Investor-Trustee Game

Reputation-formation plays a particularly critical role in the Investor-Trustee game, with even the most selfish players trying to benefit from cooperation, at least in the initial rounds. In order to reduce complexity in analyzing this, we set $\alpha_I = \beta_I = 0$ (i.e., a purely selfish Investor) and consider 2 values of $\beta_T$ (0.3 and 0.7) such that in the last round the Trustee with type $\beta_T = 0.3$ will not return any amount to the Investor and will choose fair outcome if $\beta_T = 0.7$. We generate a rich tapestry of behavior by varying the prior expectations as to $\beta_T$ and the values of strategic $(k)$ level (0,1,2) for the players.

### 3.1  Factors Affecting Behaviour

As an example, fig. 3 shows the evolution of the Players' $\mathcal{Q}$-values and 1st-order beliefs of the Investor and 2nd-order beliefs of the Trustee (i.e., her beliefs as to the Investor's beliefs about her value of $\beta_T$) over the course of a single game. Here, both players have $k_I = k_T = 1$ (i.e. they are strategic players), but the Trustee is actually less guilty $\beta_T = 0.3$.

In the first round, the Investor gives \$15, and receives back \$30 from the Trustee. This makes the Investor's beliefs about $\beta_T$ go from being uniform to being about 0.75 for $\beta_T = 0.7$ and 0.25 for $\beta_T = 0.3$ (showing the success in the Trustee's exercise in belief manipulation). This causes the $\mathcal{Q}$-value for the action corresponding to giving \$20 dollars to be highest, inspiring the Investor's generosity in round 2. Equally, the Trustee's (2nd-order) beliefs after receiving \$15 in the first round peak for the value $\beta_T = 0.7$, corresponding to thinking that the Investor believes the Trustee is Nice.

In subsequent rounds, the Trustee's nastiness limits what she returns, and so the Investor ceases giving high amounts. In response, in rounds 5 and 7, the Trustee tries to coax the Investor. We find this "reciprocal give and take" to be a characteristic behaviour of strategic Investors and Trustees (with $k = 1$). For naive Players with $k = 0$, a return of a very low amount for a high amount invested would lead to a complete breakdown of Trust formation.

Fig. 4 shows the statistics of dyadic interactions between Investors and Trustees with Uniform priors. The amount given by the Investor varies significantly depending on whether or not he is strategic, and also on his priors. In round 1, Investors with $k_I = 0$ and 1 offer \$20 first (the optimal probing action based on uniform prior beliefs) and for $k_I = 2$ offers \$15 dollars. The corresponding amount returned by the Trustee depends significantly on $k_T$. A Trustee with $k_T = 0$ and low $\beta_T$ will return nothing whereas an unconditionally cooperative Trustee (high $\beta_T$) returns roughly the same amount as received. Irrespective of the Trustee's $\beta_T$ type, the amount returned by strategic Trustees with $k_T = 1, 2$ is higher (between 1.5 and 2 times the amount received).

In round 2 we find that the low amount received causes trust to break down for Investors with $k_I = 0$. In fact, naive Investors and Trustees do not form Trust in this game. Strategic Trustees return more initially and are able to coax naive Investors to give higher amounts in the game. Generally unconditionally cooperative Trustees return more, and form Trust throughout the game if they are strategic or if they are playing against strategic Investors. Trustees with low $\beta_T$ defect towards the end of the game but coax more investment in the beginning of the game.

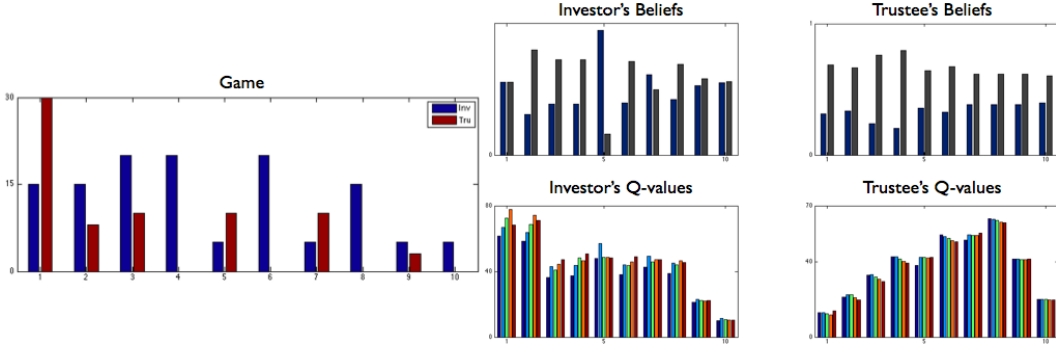

Figure 3: The generated game shows the amount given by an Investor with $k_I = 1$ and a Trustee with $\beta_T = 0.3$ and $k_T = 1$. The red bar indicates amount given by the Investor and the blue bar is the amount returned by the Trustee (after receiving 3 times amount given by the Investor). The figures on the right reveal the inner workings of the algorithm: $Q$-values through the rounds of the game for 5 different actions of the Investor (0, 5, 10, 15, 20) and 5 actions of the Trustee between values 0 and 3 times amount given by Investor. Also shown are the Investor's 1st-order beliefs (left bar for $\beta_T = 0.3$ and right bar for $\beta_T = 0.7$) and Trustee's 2nd-order beliefs over the rounds.

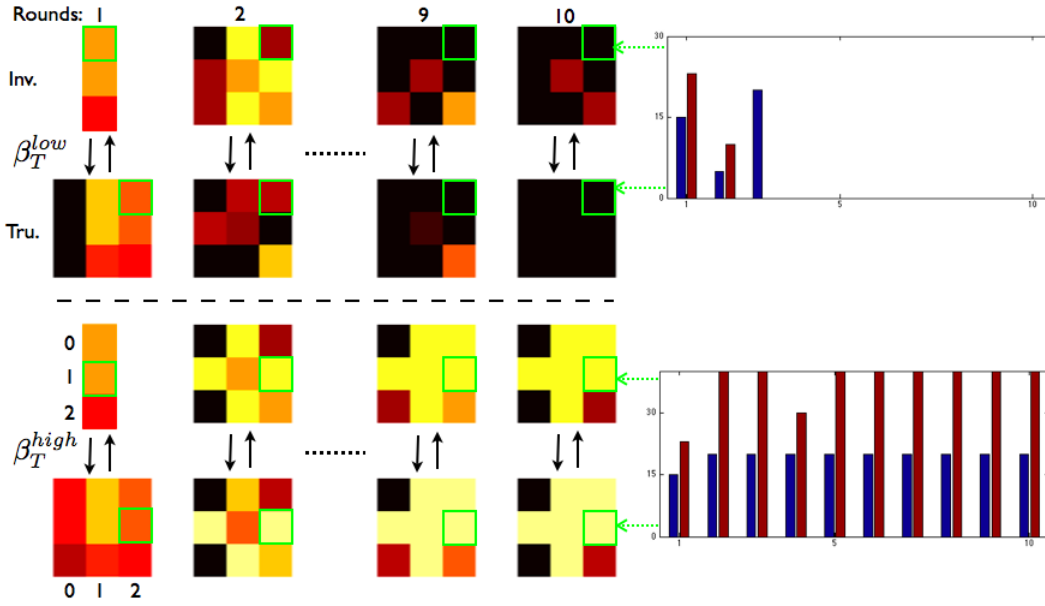

Figure 4: The dyadic interactions between the Investor and Trustee across the 10 rounds of the game. The top half shows Investor playing against Trustee with low $\beta_T$ (= 0.3) and the bottom half is the Trustee with high $\beta_T$ (= 0.7): unconditionally cooperative. The top dyad shows the amount given the Investor and the bottom dyad shows the amount returned by Trustee. Within each dyad the rows represent the strategic ($k_I$) levels of Investor (0, 1 or 2) and the columns represent $k_T$ level of the Trustee (0, 1 or 2). The dyads are shown here for the first 2 and final 2 rounds. Two particular examples are highlighted within the dyads: Investor with $k_I = 0$ and Trustee with $k_T = 2$, uncooperative ($\beta_T^{low}$) and Investor $k_I = 1$ and Trustee $k_T = 2$, cooperative ($\beta_T^{high}$). Lighter colours reveal higher amounts (with amount given by Investor in first round being 15 dollars).

The effect of strategic level is more dramatic for the Investor, since his ability to defect at any point places him in effective charge of the interaction. Strategic Investors give more money in the game than naive Investors. Consequently they also get more return on their investment because of the beneficial effects of this on their reputations. A further observation is that strategic Investors are more immune to the Trustee's actions. While this means that break-downs in the game due to

mistakes of the Trustee (or unfortunate choices from her softmax) are more easily corrected by the strategic Investor, he is also more likely to continue investing even if the Trustee doesn't reciprocate.

It is also worth noting the differences between $k = 1$ and $k = 2$ players. The latter typically offer less in the game and are also less susceptible to the actions of their opponent. Overall in this game, the Investors with $k_I = 1$ make the most amount of money playing against a cooperative Trustee while $k_I = 0$ Investors make the least. The best dyad consists of a $k_I = 1$ Investor playing with a cooperative Trustee with $k_T = 0$ or 1.

A very wide range of patterns of dyadic interaction, including the main observations of [15], can thus be captured by varying just the limited collection of parameters of our model

## 4 Recognition and Classification

One of the main reasons to build this generative model for play is to have a refined method for classifying individual players on the basis of the dyadic behaviour. We do this by considering the statistical inverse of the generative model as a recognition model. Denote the sequence of plays in the 10-round Investor-Trustee game as $\mathcal{D} = \{[a_1^{(1)}, a_2^{(1)}], .., [a_1^{(10)}, a_2^{10}]\}$. Since the game is Markovian we can calculate the probability of player $i$ taking the action sequence $\{a_i^{(t)}, t = 1, ..., 10\}$ given his Type $T_i$ and prior beliefs $\vec{b}_i^{(0)}$ as:

$$P(\{a_i^t\}|T_i, \vec{b}_i^{(0)}) = P(a_1^{(1)}|T_i, \vec{b}_i^{(0)}) \prod_{t=2}^{10} P(a_i^{(t)}|\mathcal{D}^{(t)}, T_i) \tag{6}$$

where $P(a_1^{(1)}|T_i, \vec{b}_i^{(0)})$ is the probability of initial action $a_i^{(1)}$ given by the softmax distribution and prior beliefs $\vec{b}_i^{(0)}$, and $P(a_i^{(t)}|\mathcal{D}^{(t)}, T_i)$ is the probability of action $a_i^{(t)}$ after updating beliefs $\vec{b}_i^{(t)}$ from previous beliefs $\vec{b}_i^{(t-1)}$ upon the observation of the past sequence of moves $\mathcal{D}^{(t)}$. This is a likelihood function for $T_i, \vec{b}_i^{(0)}$, and so can be used for posterior inference about type given $\mathcal{D}$. We classify the players for their utility function ($\beta_T$ value for the Trustee), strategic (ToM) levels and prior beliefs using the MAP value $(T_i^*, \vec{b}_i^{(0)*}) = max_{T_i, \vec{b}_i^{(0)}} P(\mathcal{D}|T_i, \vec{b}_i^{(0)})$.

We used our recognition model to classify subject pairs playing the 10-round Investor-Trustee game [15]. The data included 48 student pairs playing an Impersonal task for which the opponents' identities were hidden and 54 student pairs playing a Personal task for which partners met.

Each Investor-Trustee pair was classified for their level of strategic thinking $k$ and the Trustee's $\beta_T$ type (cooperative/uncooperative; see the table in Figure 5). We are able to capture some characteristic behaviours with our model. The highlighted interactions reveal that many of the pairs in the Impersonal task consisted of strategic Investors and cooperative Trustees, who formed trust in the game with the levels of investment decreasing towards the end of the game. We also highlight the difference between strategic and non-strategic Investors. An Investor with $k_I = 0$ will not form trust if the Trustee does not return a significant amount initially whilst an Investor with $k_I = 2$ will continue offering money in the game even if the Trustee gives back less than fair amounts in return. There is also a strong correlation between the proportion of Trustees classified as being cooperative: estimated as 48%, 30%, on the Impersonal and Personal tasks respectively and the corresponding Return on Investment (how much the Investor receives for the amount Invested): 120%, 109%.

Although the recognition model captures key characteristics, we do not expect the Trustees to have the specified values of $\beta_T^{low} = 0.3$ and $\beta_T^{high} = 0.7$. To test the robustness of the recognition model we generated behaviours (450 dyads) with different values of $\beta_T$ ($\beta_T^{low} = [0, 0.1, 0.2, 0.3, 0.4]$ and $\beta_T^{high} = [0.6, 0.7, 0.8, 0.9, 1.0]$), that were classified using the recognition model. Figure 5 shows how confidently players of the given type were classified to have that type.

We find that the recognition model tends to misclassify Trustees with low $\beta_T$ as having $k_T = 2$. This is because the Trustees with those characteristics will offer high amounts to coax the Investor. Investor are shown to be correctly classified in most cases. Overall the recognition model has a tendency to assign higher $k_T$ to the Trustees than their true type, though the model correctly assigns the right cooperative/uncooperative type to the Trustee.

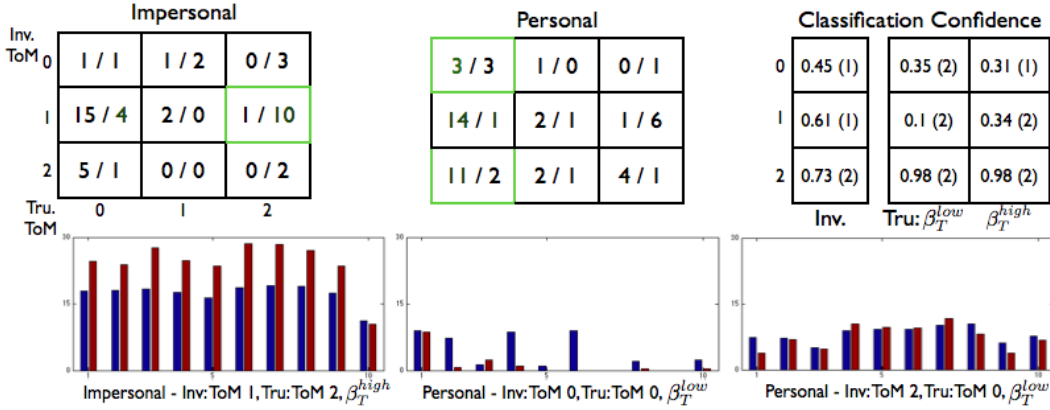

Figure 5: Subject pairs are classified into levels of Theory of Mind for the Investor (rows) and Trustee (columns). The number of subject-pairs with the classification are shown in each entry along with whether the Trustee was classified as uncooperative / cooperative ($\beta_T^{low}$, $\beta_T^{high}$). The subjects play an Impersonal game where they do not know the identities of the opponent and a Personal game where identities are revealed.

We reveal the dominant or unique behavioural classification within tables (highlighted): Impersonal ($k_I = 1$, $k_T = 2$, cooperative) group averaged over 10 subjects, Personal group ($k_I = 0$, $k_T = 0$, uncooperative) averaged over 3 subjects, and Personal group with ($k_I = 2$, $k_T = 0$, uncooperative) averaged over 11 subjects.

We also show the classification confidence for the types given the behaviour was generated from our model with other values of $\beta_T$ for the Trustee, as well as the type that the player is most likely to be classified as in brackets. (A Trustee with low $\beta_T$ and $k_T = 1$ is very likely to be misclassified as a player with $k_T = 2$, while a player with $k_T = 2$ will mostly be classified with $k_T = 2$)

## 5   Discussion

We built a generative model that captures classes of observed behavior in multi-round trust tasks. The critical features of the model are a social utility function, with parameters covering different types of subjects; partial observability, accounting for subjects' ignorance about their opponents; an explicit and finite cognitive hierarchy to make approximate equilibrium calculations marginally tractable; and partly deterministic and partly sample-based evaluation methods.

Despite its descriptive adequacy, we do not claim that it is uniquely competent. We also do not suggest a normative rationale for pieces of the model such as the social utility function. Nevertheless, the separation between the vagaries of utility and the exactness of inference is attractive, not the least by providing clearly distinct signals as to the inner workings of the algorithm that can be extremely useful to capture neural findings. Indeed, the model is relevant to a number of experimental findings, including those due to [15], [18], [19]. The underlying foundation in reinforcement learning is congenial, given the substantial studies of the neural bases of this [20].

The model does directly license some conclusions. For instance, we postulate that higher activation will be observed in regions of the brain associated with theory of mind for Investors that give more in the game, and for Trustees that can coax more. However, unlike [13] our Naive players still build models, albeit unsophisticated ones, of the other player (in contrast to level 0 players who assume the opponent to play a random strategy). So this might lead to an investigation of how sophisticated and naive theory of mind models are built by subjects in the game.

We also constructed the recognition model, which is the statistical inverse to this generative model. While we showed this to capture a broad class of behaviours, it only explains the coarse features of the behaviour. We need to incorporate some of the other parameters of our model, such as the Investor's *envy* and the temperature parameter of the softmax distribution in order to capture the nuances in the interactions. Further it would be interesting to use the recognition model in pathological populations, looking at such conditions as autism and borderline personality disorder.

Finally, this computational model provides a guide for designing experiments to probe aspects of social utility, strategic thinking levels and prior beliefs, as well as inviting ready extensions to related tasks such as Public Goods games. The inference method may also have wider application, for instance to identifying which of a collection of Bayes-Nash equilibria is most likely to arise, given psychological factors about human utilities.

**Acknowledgments**

We thank Wako Yoshida, Karl Friston and Terry Lohrenz for useful discussions.

## References

[1] K.A. McCabe, M.L. Rigdon and V.L. Smith. Positive Reciprocity and Intentions in Trust Games (2003). Journal of Economic Behaviour and Organization.

[2] E. Fehr and K.M. Schmidt. A Theory of Fairness, Competition and Cooperation (1999). The Quarterly Journal of Economics.

[3] E. Fehr and S. Gachter. Fairness and Retaliation: The Economics of Reciprocity (2000). Journal of Economic Perspectives.

[4] E. Fehr and U. Fischbacher. Social norms and human cooperation (2004). *TRENDS in Cog. Sci.* 8:4.

[5] P.J. Gmytrasiewicz and P. Doshi. A Framework for Sequential Planning in Multi-Agent Settings (2005). Journal of Artificial Intelligence Research.

[6] V. Conitzer and T. Sandholm (2002). Complexity Results about Nash Equilibria. Technical Report CMU-CS-02-135, School of Computer Science, Carnegie-Mellon University.

[7] S. Thrun. Monte Carlo POMDPs (2000). Advances in Neural Information Processing Systems 12.

[8] JC Harsanyi (1967). Games with Incomplete Information Played by "Bayesian" Players, I-III. Management Science.

[9] J.F. Mertens and S. Zamir. Formulation of Bayesian analysis for games with incomplete information (1985). International Journal of Game Theory.

[10] Y. Nyarko. Convergence in Economic Models with Bayesian Hierarchies of Beliefs (1997). Journal of Economic Theory.

[11] C. Camerer. Behavioural Game Theory: Experiments in Strategic Interaction (2003). Princeton Univ.

[12] R. McKelvey and T. Palfrey. Quantal Response Equilibria for Extensive Form Games (1998). Experimental Economics 1:9-41.

[13] C. Camerer, T-H. Ho and J-K. Chong. A Cognitive Hierarchy Model of Games (2004). The Quarterly Journal of Economics.

[14] R.D. McKelvey, A.M. McLennan and T.L. Turocy (2007). Gambit: Software Tools for Game Theory.

[15] B. King-Casas, D. Tomlin, C. Anen, C.F. Camerer, S.R. Quartz and P.R. Montague (2005). Getting to know you: Reputation and Trust in a two-person economic exchange. *Science* 308:78-83.

[16] D. Tomlin, M.A. Kayali, B. King-Casas, C. Anen, C.F. Camerer, S.R. Quartz and P.R. Montague (2006). Agent-specific responses in cingulate cortex during economic exchanges. *Science* 312:1047-1050.

[17] L.P. Kaelbling, M.L. Littman and A.R. Cassandra. Planning and acting in partially observable stochastic domains (1998). Artificial Intelligence.

[18] K. McCabe, D. Houser, L. Ryan, V. Smith, T. Trouard. A functional imaging study of cooperation in two-person reciprocal exchange. *Proc. Natl. Acad. Sci. USA* 98:11832-35.

[19] K. Fliessbach, B. Weber, P. Trautner, T. Dohmen, U. Sunde, C.E. Elger and A. Falk. Social Comparison Affects Reward-Related Brain Activity in the Human Ventral Striatum (2007). *Science* 318:1302-1305.

[20] B. Lau and P. W. Glimcher (2008). Representations in the Primate Striatum during Matching Behaviour. Neuron 58.

